# Hebbian Learning of Bayes Optimal Decisions

**Bernhard Nessler,**[*] **Michael Pfeiffer,**[*] **and Wolfgang Maass**
Institute for Theoretical Computer Science
Graz University of Technology
A-8010 Graz, Austria
{nessler,pfeiffer,maass}@igi.tugraz.at

## Abstract

Uncertainty is omnipresent when we perceive or interact with our environment, and the Bayesian framework provides computational methods for dealing with it. Mathematical models for Bayesian decision making typically require data-structures that are hard to implement in neural networks. This article shows that even the simplest and experimentally best supported type of synaptic plasticity, Hebbian learning, in combination with a sparse, redundant neural code, can in principle learn to infer optimal Bayesian decisions. We present a concrete Hebbian learning rule operating on log-probability ratios. Modulated by reward-signals, this Hebbian plasticity rule also provides a new perspective for understanding how Bayesian inference could support fast reinforcement learning in the brain. In particular we show that recent experimental results by Yang and Shadlen [1] on reinforcement learning of probabilistic inference in primates can be modeled in this way.

## 1  Introduction

Evolution is likely to favor those biological organisms which are able to maximize the chance of achieving correct decisions in response to multiple unreliable sources of evidence. Hence one may argue that probabilistic inference, rather than logical inference, is the "mathematics of the mind", and that this perspective may help us to understand the principles of computation and learning in the brain [2]. Bayesian inference, or equivalently inference in Bayesian networks [3] is the most commonly considered framework for probabilistic inference, and a mathematical theory for learning in Bayesian networks has been developed.

Various attempts to relate these theoretically optimal models to experimentally supported models for computation and plasticity in networks of neurons in the brain have been made. [2] models Bayesian inference through an approximate implementation of the Belief Propagation algorithm (see [3]) in a network of spiking neurons. For reduced classes of probability distributions, [4] proposed a method for spiking network models to learn Bayesian inference with an online approximation to an EM algorithm. The approach of [5] interprets the weight $w_{ji}$ of a synaptic connection between neurons representing the random variables $x_i$ and $x_j$ as $\log \frac{p(x_i,x_j)}{p(x_i) \cdot p(x_j)}$, and presents algorithms for learning these weights.

Neural correlates of variables that are important for decision making under uncertainty had been presented e.g. in the recent experimental study by Yang and Shadlen [1]. In their study they found that firing rates of neurons in area LIP of macaque monkeys reflect the log-likelihood ratio (or log-odd) of the outcome of a binary decision, given visual evidence. The learning of such log-odds for Bayesian decision making can be reduced to learning weights for a linear classifier, given an appropriate but fixed transformation from the input to possibly nonlinear features [6]. We show

---

[*]Both authors contributed equally to this work.

that the optimal weights for the linear decision function are actually log-odds themselves, and the definition of the features determines the assumptions of the learner about statistical dependencies among inputs.

In this work we show that simple Hebbian learning [7] is sufficient to implement learning of Bayes optimal decisions for arbitrarily complex probability distributions. We present and analyze a concrete learning rule, which we call the *Bayesian Hebb rule*, and show that it provably converges towards correct log-odds. In combination with appropriate preprocessing networks this implements learning of different probabilistic decision making processes like e.g. Naive Bayesian classification. Finally we show that a reward-modulated version of this Hebbian learning rule can solve simple reinforcement learning tasks, and also provides a model for the experimental results of [1].

## 2   A Hebbian rule for learning log-odds

We consider the model of a linear threshold neuron with output $y_0$, where $y_0 = 1$ means that the neuron is firing and $y_0 = 0$ means non-firing. The neuron's current decision $\hat{y_0}$ whether to fire or not is given by a linear decision function $\hat{y_0} = \text{sign}(w_0 \cdot constant + \sum_{i=1}^{n} w_i y_i)$, where the $y_i$ are the current firing states of all presynaptic neurons and $w_i$ are the weights of the corresponding synapses.

We propose the following learning rule, which we call the Bayesian Hebb rule:

$$
\boxed{
\Delta w_i = \begin{cases}
\eta \left(1 + e^{-w_i}\right), & \text{if } y_0 = 1 \text{ and } y_i = 1 \\
-\eta \left(1 + e^{w_i}\right), & \text{if } y_0 = 0 \text{ and } y_i = 1 \\
0, & \text{if } y_i = 0.
\end{cases}
}
\tag{1}
$$

This learning rule is purely local, i.e. it depends only on the binary firing state of the pre- and postsynaptic neuron $y_i$ and $y_0$, the current weight $w_i$ and a learning rate $\eta$. Under the assumption of a stationary joint probability distribution of the pre- and postsynaptic firing states $y_0, y_1, \ldots, y_n$ the Bayesian Hebb rule learns log-probability ratios of the postsynaptic firing state $y_0$, conditioned on a corresponding presynaptic firing state $y_i$. We consider in this article the use of the rule in a supervised, teacher forced mode (see Section 3), and also in a reinforcement learning mode (see Section 4). We will prove that the rule converges globally to the target weight value $w_i^*$, given by

$$
w_i^* = \log \frac{p(y_0 = 1 | y_i = 1)}{p(y_0 = 0 | y_i = 1)} \quad .
\tag{2}
$$

We first show that the expected update $\mathrm{E}[\Delta w_i]$ under (1) vanishes at the target value $w_i^*$:

$$
\mathrm{E}[\Delta w_i^*] = 0 \;\Leftrightarrow\; p(y_0{=}1, y_i{=}1)\eta(1 + e^{-w_i^*}) - p(y_0{=}0, y_i{=}1)\eta(1 + e^{w_i^*}) = 0
$$

$$
\Leftrightarrow \qquad \frac{1 + e^{w_i^*}}{1 + e^{-w_i^*}} = \frac{p(y_0{=}1, y_i{=}1)}{p(y_0{=}0, y_i{=}1)}
$$

$$
\Leftrightarrow \qquad w_i^* = \log \frac{p(y_0{=}1 | y_i{=}1)}{p(y_0{=}0 | y_i{=}1)} \quad .
\tag{3}
$$

Since the above is a chain of equivalence transformations, this proves that $w_i^*$ is the only equilibrium value of the rule. The weight vector $\mathbf{w}^*$ is thus a global point-attractor with regard to expected weight changes of the Bayesian Hebb rule (1) in the $n$-dimensional weight-space $\mathbb{R}^n$.

Furthermore we show, using the result from (3), that the expected weight change at any current value of $w_i$ points in the direction of $w_i^*$. Consider some arbitrary intermediate weight value $w_i = w_i^* + 2\epsilon$:

$$
\begin{aligned}
\mathrm{E}[\Delta w_i]|_{w_i^* + 2\epsilon} &= \mathrm{E}[\Delta w_i]|_{w_i^* + 2\epsilon} - E[\Delta w_i]|_{w_i^*} \\
&\propto p(y_0{=}1, y_i{=}1)e^{-w_i^*}(e^{-2\epsilon} - 1) - p(y_0{=}0, y_i{=}1)e^{w_i^*}(e^{2\epsilon} - 1) \\
&= (p(y_0{=}0, y_i{=}1)e^{-\epsilon} + p(y_0{=}1, y_i{=}1)e^{\epsilon})(e^{-\epsilon} - e^{\epsilon}) \quad .
\end{aligned}
\tag{4}
$$

The first factor in (4) is always non-negative, hence $\epsilon < 0$ implies $E[\Delta w_i] > 0$, and $\epsilon > 0$ implies $E[\Delta w_i] < 0$. The Bayesian Hebb rule is therefore always expected to perform updates in the right direction, and the initial weight values or perturbations of the weights decay exponentially fast.

Already after having seen a finite set of examples $\langle y_0, \ldots, y_n \rangle \in \{0, 1\}^{n+1}$, the Bayesian Hebb rule closely approximates the optimal weight vector $\hat{\mathbf{w}}$ that can be inferred from the data. A traditional frequentist's approach would use counters $a_i = \#[y_0{=}1 \wedge y_i{=}1]$ and $b_i = \#[y_0{=}0 \wedge y_i{=}1]$ to estimate every $w_i^*$ by

$$\hat{w}_i = \log \frac{a_i}{b_i} . \tag{5}$$

A Bayesian approach would model $p(y_0|y_i)$ with an (initially flat) $Beta$-distribution, and use the counters $a_i$ and $b_i$ to update this belief [3], leading to the same MAP estimate $\hat{w}_i$. Consequently, in both approaches a new example with $y_0 = 1$ and $y_i = 1$ leads to the update

$$\hat{w}_i^{new} = \log \frac{a_i + 1}{b_i} = \log \frac{a_i}{b_i} \left( 1 + \frac{1}{a_i} \right) = \hat{w}_i + \log(1 + \frac{1}{N_i}(1 + e^{-\hat{w}_i})) , \tag{6}$$

where $N_i := a_i + b_i$ is the number of previously processed examples with $y_i = 1$, thus $\frac{1}{a_i} = \frac{1}{N_i}(1 + \frac{b_i}{a_i})$. Analogously, a new example with $y_0 = 0$ and $y_i = 1$ gives rise to the update

$$\hat{w}_i^{new} = \log \frac{a_i}{b_i + 1} = \log \frac{a_i}{b_i} \left( \frac{1}{1 + \frac{1}{b_i}} \right) = \hat{w}_i - \log(1 + \frac{1}{N_i}(1 + e^{\hat{w}_i})). \tag{7}$$

Furthermore, $\hat{w}_i^{new} = \hat{w}_i$ for a new example with $y_i = 0$. Using the approximation $\log(1 + \alpha) \approx \alpha$ the update rules (6) and (7) yield the Bayesian Hebb rule (1) with an adaptive learning rate $\eta_i = \frac{1}{N_i}$ for each synapse.

In fact, a result of Robbins-Monro (see [8] for a review) implies that the updating of weight estimates $\hat{w}_i$ according to (6) and (7) converges to the target values $w_i^*$ not only for the particular choice $\eta_i^{(N_i)} = \frac{1}{N_i}$, but for any sequence $\eta_i^{(N_i)}$ that satisfies $\sum_{N_i=1}^{\infty} \eta_i^{(N_i)} = \infty$ and $\sum_{N_i=1}^{\infty} (\eta_i^{(N_i)})^2 < \infty$. More than that the Supermartingale Convergence Theorem (see [8]) guarantees convergence in distribution even for a sufficiently small constant learning rate.

**Learning rate adaptation**

One can see from the above considerations that the Bayesian Hebb rule with a constant learning rate $\eta$ converges globally to the desired log-odds. A too small constant learning rate, however, tends to slow down the initial convergence of the weight vector, and a too large constant learning rate produces larger fluctuations once the steady state is reached.

(6) and (7) suggest a decaying learning rate $\eta_i^{(N_i)} = \frac{1}{N_i}$, where $N_i$ is the number of preceding examples with $y_i = 1$. We will present a learning rate adaptation mechanism that avoids biologically implausible counters, and is robust enough to deal even with non-stationary distributions.

Since the Bayesian Hebb rule and the Bayesian approach of updating $Beta$-distributions for conditional probabilities are closely related, it is reasonable to expect that the distribution of weights $w_i$ over longer time periods with a non-vanishing learning rate will resemble a $Beta(a_i, b_i)$-distribution transformed to the log-odd domain. The parameters $a_i$ and $b_i$ in this case are not exact counters anymore but correspond to virtual sample sizes, depending on the current learning rate. We formalize this statistical model of $w_i$ by

$$\sigma(w_i) = \frac{1}{1 + e^{-w_i}} \sim Beta(a_i, b_i) \iff w_i \sim \frac{\Gamma(a_i + b_i)}{\Gamma(a_i)\Gamma(b_i)} \sigma(w_i)^{a_i} \sigma(-w_i)^{b_i},$$

In practice this model turned out to capture quite well the actually observed quasi-stationary distribution of $w_i$. In [9] we show analytically that $\mathrm{E}[w_i] \approx \log \frac{a_i}{b_i}$ and $\mathrm{Var}[w_i] \approx \frac{1}{a_i} + \frac{1}{b_i}$. A learning rate adaptation mechanism at the synapse that keeps track of the observed mean and variance of the synaptic weight can therefore recover estimates of the virtual sample sizes $a_i$ and $b_i$. The following mechanism, which we call *variance tracking* implements this by computing running averages of the weights and the squares of weights in $\bar{w}_i$ and $\bar{q}_i$:

$$\begin{aligned} \eta_i^{new} &\leftarrow \frac{\bar{q}_i - \bar{w}_i^2}{1 + \cosh \bar{w}_i} \\ \bar{w}_i^{new} &\leftarrow (1 - \eta_i) \bar{w}_i + \eta_i w_i \\ \bar{q}_i^{new} &\leftarrow (1 - \eta_i) \bar{q}_i + \eta_i w_i^2 \quad . \end{aligned} \tag{8}$$

In practice this mechanism decays like $\frac{1}{N_i}$ under stationary conditions, but is also able to handle changing input distributions. It was used in all presented experiments for the Bayesian Hebb rule.

## 3  Hebbian learning of Bayesian decisions

We now show how the Bayesian Hebb rule can be used to learn Bayes optimal decisions. The first application is the Naive Bayesian classifier, where a binary target variable $x_0$ should be inferred from a vector of multinomial variables $\mathbf{x} = \langle x_1, \ldots, x_m \rangle$, under the assumption that the $x_i$'s are conditionally independent given $x_0$, thus $p(x_0, \mathbf{x}) = p(x_0) \prod_1^m p(x_k | x_0)$. Using basic rules of probability theory the posterior probability ratio for $x_0 = 1$ and $x_0 = 0$ can be derived:

$$\frac{p(x_0{=}1|\mathbf{x})}{p(x_0{=}0|\mathbf{x})} = \frac{p(x_0{=}1)}{p(x_0{=}0)} \prod_{k=1}^m \frac{p(x_k|x_0{=}1)}{p(x_k|x_0{=}0)} = \left(\frac{p(x_0{=}1)}{p(x_0{=}0)}\right)^{(1-m)} \prod_{k=1}^m \frac{p(x_0{=}1|x_k)}{p(x_0{=}0|x_k)} = \quad (9)$$

$$= \left(\frac{p(x_0{=}1)}{p(x_0{=}0)}\right)^{(1-m)} \prod_{k=1}^m \prod_{j=1}^{m_k} \left(\frac{p(x_0{=}1|x_k{=}j)}{p(x_0{=}0|x_k{=}j)}\right)^{I(x_k=j)} ,$$

where $m_k$ is the number of different possible values of the input variable $x_k$, and the indicator function $I$ is defined as $I(true) = 1$ and $I(false) = 0$.

Let the $m$ input variables $x_1, \ldots, x_m$ be represented through the binary firing states $y_1, \ldots, y_n \in \{0, 1\}$ of the $n$ presynaptic neurons in a population coding manner. More precisely, let each input variable $x_k \in \{1, \ldots, m_k\}$ be represented by $m_k$ neurons, where each neuron fires only for one of the $m_k$ possible values of $x_k$. Formally we define the simple preprocessing (*SP*)

$$\mathbf{y}^\mathsf{T} = \left[\phi(x_1)^\mathsf{T}, \ldots, \phi(x_m)^\mathsf{T}\right] \quad \text{with} \quad \phi(x_k)^\mathsf{T} = [I(x_k = 1), \ldots, I(x_k = m_k)] . \quad (10)$$

The binary target variable $x_0$ is represented directly by the binary state $y_0$ of the postsynaptic neuron. Substituting the state variables $y_0, y_1, \ldots, y_n$ in (9) and taking the logarithm leads to

$$\log \frac{p(y_0 = 1|\mathbf{y})}{p(y_0 = 0|\mathbf{y})} = (1 - m) \log \frac{p(y_0 = 1)}{p(y_0 = 0)} + \sum_{i=1}^n y_i \log \frac{p(y_i = 1|y_0 = 1)}{p(y_i = 1|y_0 = 0)} .$$

Hence the optimal decision under the Naive Bayes assumption is

$$\hat{y}_0 = \text{sign}((1 - m)w_0^* + \sum_{i=1}^n w_i^* \, y_i) \quad .$$

The optimal weights $w_0^*$ and $w_i^*$

$$w_0^* = \log \frac{p(y_0 = 1)}{p(y_0 = 0)} \quad \text{and} \quad w_i^* = \log \frac{p(y_0 = 1|y_i = 1)}{p(y_0 = 0|y_i = 1)} \quad \text{for} \quad i = 1, \ldots, n.$$

are obviously log-odds which can be learned by the Bayesian Hebb rule (the bias weight $w_0$ is simply learned as an unconditional log-odd).

### 3.1  Learning Bayesian decisions for arbitrary distributions

We now address the more general case, where conditional independence of the input variables $x_1, \ldots, x_m$ cannot be assumed. In this case the dependency structure of the underlying distribution is given in terms of an arbitrary Bayesian network BN for discrete variables (see e.g. Figure 1 A). Without loss of generality we choose a numbering scheme of the nodes of the BN such that the node to be learned is $x_0$ and its direct children are $x_1, \ldots, x_{m'}$. This implies that the BN can be described by $m + 1$ (possibly empty) parent sets defined by

$$\mathsf{P}_k = \{i \mid \text{a directed edge } x_i \rightarrow x_k \text{ exists in BN and } i \geq 1\} \quad .$$

The joint probability distribution on the variables $x_0, \ldots, x_m$ in BN can then be factored and evaluated for $x_0 = 1$ and $x_0 = 0$ in order to obtain the probability ratio

$$\frac{p(x_0 = 1, \mathbf{x})}{p(x_0 = 0, \mathbf{x})} = \frac{p(x_0 = 1|\mathbf{x})}{p(x_0 = 0|\mathbf{x})} = \frac{p(x_0 = 1|\mathbf{x}_{\mathsf{P}_0})}{p(x_0 = 0|\mathbf{x}_{\mathsf{P}_0})} \prod_{k=1}^{m'} \frac{p(x_k|\mathbf{x}_{\mathsf{P}_k}, x_0 = 1)}{p(x_k|\mathbf{x}_{\mathsf{P}_k}, x_0 = 0)} \prod_{k=m'+1}^m \frac{p(x_k|\mathbf{x}_{\mathsf{P}_k})}{p(x_k|\mathbf{x}_{\mathsf{P}_k})} \quad .$$

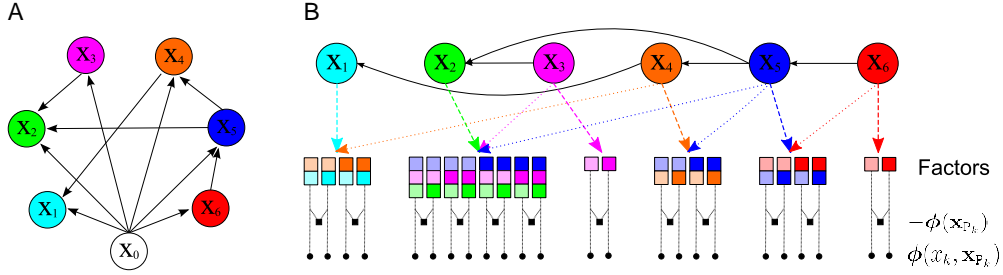

Figure 1: **A)** An example Bayesian network with general connectivity. **B)** Population coding applied to the Bayesian network shown in panel A. For each combination of values of the variables $\{x_k, \mathbf{x}_{P_k}\}$ of a factor there is exactly one neuron (indicated by a black circle) associated with the factor that outputs the value 1. In addition OR's of these values are computed (black squares). We refer to the resulting preprocessing circuit as generalized preprocessing (GP).

Obviously, the last term cancels out, and by applying Bayes' rule and taking the logarithm the target log-odd can be expressed as a sum of conditional log-odds only:

$$\log \frac{p(x_0{=}1|\mathbf{x})}{p(x_0{=}0|\mathbf{x})} = \log \frac{p(x_0{=}1|\mathbf{x}_{P_0})}{p(x_0{=}0|\mathbf{x}_{P_0})} + \sum_{k=1}^{m'} \left( \log \frac{p(x_0{=}1|x_k,\mathbf{x}_{P_k})}{p(x_0{=}0|x_k,\mathbf{x}_{P_k})} - \log \frac{p(x_0{=}1|\mathbf{x}_{P_k})}{p(x_0{=}0|\mathbf{x}_{P_k})} \right). \quad (11)$$

We now develop a suitable sparse encoding of of $x_1, \ldots, x_m$ into binary variables $y_1, \ldots, y_n$ (with $n \gg m$) such that the decision function (11) can be written as a weighted sum, and the weights correspond to conditional log-odds of $y_i$'s. Figure 1 B illustrates such a sparse code: One binary variable is created for every possible value assignment to a variable and all its parents, and one additional binary variable is created for every possible value assignment to the parent nodes only. Formally, the previously introduced population coding operator $\phi$ is generalized such that $\phi(x_{i_1}, x_{i_2}, \ldots, x_{i_l})$ creates a vector of length $\prod_{j=1}^{l} m_{i_j}$ that equals zero in all entries except for one 1-entry which identifies by its position in the vector the present assignment of the input variables $x_{i_1}, \ldots, x_{i_l}$. The concatenation of all these population coded groups is collected in the vector $\mathbf{y}$ of length $n$

$$\mathbf{y}^{\mathsf{T}} = \left[ \phi(\mathbf{x}_{P_0})^{\mathsf{T}}, \phi(x_1, \mathbf{x}_{P_1})^{\mathsf{T}}, -\phi(\mathbf{x}_{P_1})^{\mathsf{T}}, \ldots, \phi(x_m, \mathbf{x}_{P_m})^{\mathsf{T}}, -\phi(\mathbf{x}_{P_m})^{\mathsf{T}} \right] \quad . \quad (12)$$

The negated vector parts in (12) correspond to the negative coefficients in the sum in (11). Inserting the sparse coding (12) into (11) allows writing the Bayes optimal decision function (11) as a pure sum of log-odds of the target variable:

$$\hat{x_0} = \hat{y_0} = \text{sign}(\sum_{i=1}^{n} w_i^* y_i), \qquad \text{with} \qquad w_i^* = \log \frac{p(y_0{=}1|y_i{\neq}0)}{p(y_0{=}0|y_i{\neq}0)} \quad .$$

Every synaptic weight $w_i$ can be learned efficiently by the Bayesian Hebb rule (1) with the formal modification that the update is not only triggered by $y_i{=}1$ but in general whenever $y_i{\neq}0$ (which obviously does not change the behavior of the learning process). A neuron that learns with the Bayesian Hebb rule on inputs that are generated by the generalized preprocessing (*GP*) defined in (12) therefore approximates the Bayes optimal decision function (11), and converges quite fast to the best performance that any probabilistic inference could possibly achieve (see Figure 2B).

## 4 The Bayesian Hebb rule in reinforcement learning

We show in this section that a reward-modulated version of the Bayesian Hebb rule enables a learning agent to solve simple reinforcement learning tasks. We consider the standard operant conditioning scenario, where the learner receives at each trial an input $\mathbf{x} = \langle x_1, \ldots, x_m \rangle$, chooses an action $\alpha$ out of a set of possible actions $A$, and receives a binary reward signal $r \in \{0, 1\}$ with probability $p(r|\mathbf{x}, a)$. The learner's goal is to learn (as fast as possible) a policy $\pi(\mathbf{x}, a)$ so that action selection according to this policy maximizes the average reward. In contrast to the previous

learning tasks, the learner has to explore different actions for the same input to learn the reward-probabilities for all possible actions. The agent might for example choose actions stochastically with $\pi(\mathbf{x}, a = \alpha) = p(r = 1|\mathbf{x}, a = \alpha)$, which corresponds to the *matching behavior* phenomenon often observed in biology [10]. This policy was used during training in our computer experiments.

The goal is to infer the probability of binary reward, so it suffices to learn the log-odds $log \frac{p(r=1|\mathbf{x},a)}{p(r=0|\mathbf{x},a)}$ for every action, and choose the action that is most likely to yield reward (e.g. by a Winner-Take-All structure). If the reward probability for an action $a = \alpha$ is defined by some Bayesian network BN, one can rewrite this log-odd as

$$\log \frac{p(r = 1|\mathbf{x}, a = \alpha)}{p(r = 0|\mathbf{x}, a = \alpha)} = \log \frac{p(r = 1|a = \alpha)}{p(r = 0|a = \alpha)} + \sum_{k=1}^{m} \log \frac{p(x_k|\mathbf{x}_{\mathrm{P}_k}, r = 1, a = \alpha)}{p(x_k|\mathbf{x}_{\mathrm{P}_k}, r = 0, a = \alpha)}. \qquad (13)$$

In order to use the Bayesian Hebb rule, the input vector $\mathbf{x}$ is preprocessed to obtain a binary vector $\mathbf{y}$. Both a simple population code such as (10), or generalized preprocessing as in (12) and Figure 1B can be used, depending on the assumed dependency structure. The reward log-odd (13) for the preprocessed input vector $\mathbf{y}$ can then be written as a linear sum

$$\log \frac{p(r = 1|\mathbf{y}, a = \alpha)}{p(r = 0|\mathbf{y}, a = \alpha)} \quad = \quad w_{\alpha,0}^* + \sum_{i=1}^{n} w_{\alpha,i}^* \, y_i \quad ,$$

where the optimal weights are $w_{\alpha,0}^* = \log \frac{p(r=1|a=\alpha)}{p(r=0|a=\alpha)}$ and $w_{\alpha,i}^* = \log \frac{p(r=1|y_i \neq 0, a=\alpha)}{p(r=0|y_i \neq 0, a=\alpha)}$. These log-odds can be learned for each possible action $\alpha$ with a reward-modulated version of the Bayesian Hebb rule (1):

$$\Delta w_{\alpha,i} = \begin{cases} \eta \cdot (1 + e^{-w_{\alpha,i}}), & \text{if } r = 1, y_i \neq 0, a = \alpha \\ -\eta \cdot (1 + e^{w_{\alpha,i}}), & \text{if } r = 0, y_i \neq 0, a = \alpha \\ 0, & \text{otherwise} \end{cases} \qquad (14)$$

The attractive theoretical properties of the Bayesian Hebb rule for the prediction case apply also to the case of reinforcement learning. The weights corresponding to the optimal policy are the only equilibria under the reward-modulated Bayesian Hebb rule, and are also global attractors in weight space, independently of the exploration policy (see [9]).

## 5 Experimental Results

### 5.1 Results for prediction tasks

We have tested the Bayesian Hebb rule on 400 different prediction tasks, each of them defined by a general (non-Naive) Bayesian network of 7 binary variables. The networks were randomly generated by the algorithm of [11]. From each network we sampled 2000 training and 5000 test examples, and measured the percentage of correct predictions after every update step.

The performance of the predictor was compared to the Bayes optimal predictor, and to online logistic regression, which fits a linear model by gradient descent on the cross-entropy error function. This non-Hebbian learning approach is in general the best performing online learning approach for linear discriminators [3]. Figure 2A shows that the Bayesian Hebb rule with the simple preprocessing (10) generalizes better from a few training examples, but is outperformed by logistic regression in the long run, since the Naive Bayes assumption is not met. With the generalized preprocessing (12), the Bayesian Hebb rule learns fast and converges to the Bayes optimum (see Figure 2B). In Figure 2C we show that the Bayesian Hebb rule is robust to noisy updates - a condition very likely to occur in biological systems. We modified the weight update $\Delta w_i$ such that it was uniformly distributed in the interval $\Delta w_i \pm \gamma\%$. Even such imprecise implementations of the Bayesian Hebb rule perform very well. Similar results can be obtained if the $\exp$-function in (1) is replaced by a low-order Taylor approximation.

### 5.2 Results for action selection tasks

The reward-modulated version (14), of the Bayesian Hebb rule was tested on 250 random action selection tasks with $m = 6$ binary input attributes, and 4 possible actions. For every action a

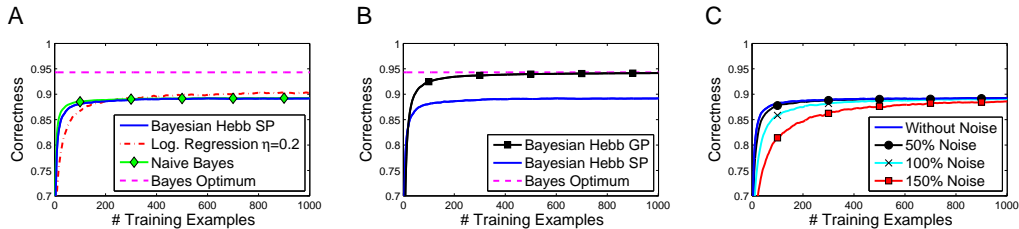

Figure 2: Performance comparison for prediction tasks. **A)** The Bayesian Hebb rule with simple preprocessing (*SP*) learns as fast as Naive Bayes, and faster than logistic regression (with optimized constant learning rate). **B)** The Bayesian Hebb rule with generalized preprocessing (*GP*) learns fast and converges to the Bayes optimal prediction performance. **C)** Even a very imprecise implementation of the Bayesian Hebb rule (noisy updates, uniformly distributed in $\Delta w_i \pm \gamma\%$) yields almost the same learning performance.

random Bayesian network [11] was drawn to model the input and reward distributions (see [9] for details). The agent received stochastic binary rewards for every chosen action, updated the weights $w_{\alpha,i}$ according to (14), and measured the average reward on 500 independent test trials.

In Figure 3A we compare the reward-modulated Bayesian Hebb rule with simple population coding (10) (*Bayesian Hebb SP*), and generalized preprocessing (12) (*Bayesian Hebb GP*), to the standard learning model for simple conditioning tasks, the non-Hebbian Rescorla-Wagner rule [12]. The reward-modulated Bayesian Hebb rule learns as fast as the Rescorla-Wagner rule, and achieves in combination with generalized preprocessing a higher performance level. The widely used tabular Q-learning algorithm, in comparison is slower than the other algorithms, since it does not generalize, but it converges to the optimal policy in the long run.

## 5.3   A model for the experiment of Yang and Shadlen

In the experiment by Yang and Shadlen [1], a monkey had to choose between gazing towards a red target $R$ or a green target $G$. The probability that a reward was received at either choice depended on four visual input stimuli that had been shown at the beginning of the trial. Every stimulus was one shape out of a set of ten possibilities and had an associated weight, which had been defined by the experimenter. The sum of the four weights yielded the log-odd of obtaining a reward at the red target, and a reward for each trial was assigned accordingly to one of the targets. The monkey thus had to combine the evidence from four visual stimuli to optimize its action selection behavior.

In the model of the task it is sufficient to learn weights only for the action $a = R$, and select this action whenever the log-odd using the current weights is positive, and $G$ otherwise. A simple population code as in (10) encoded the 4-dimensional visual stimulus into a 40-dimensional binary vector $\mathbf{y}$. In our experiments, the reward-modulated Bayesian Hebb rule learns this task as fast and with similar quality as the non-Hebbian Rescorla-Wagner rule. Furthermore Figures 3B and 3C show that it produces after learning similar behavior as that reported for two monkeys in [1].

## 6   Discussion

We have shown that the simplest and experimentally best supported local learning mechanism, Hebbian learning, is sufficient to learn Bayes optimal decisions. We have introduced and analyzed the Bayesian Hebb rule, a training method for synaptic weights, which converges fast and robustly to optimal log-probability ratios, without requiring any communication between plasticity mechanisms for different synapses. We have shown how the same plasticity mechanism can learn Bayes optimal decisions under different statistical independence assumptions, if it is provided with an appropriately preprocessed input. We have demonstrated on a variety of prediction tasks that the Bayesian Hebb rule learns very fast, and with an appropriate sparse preprocessing mechanism for groups of statistically dependent features its performance converges to the Bayes optimum. Our approach therefore suggests that sparse, redundant codes of input features may simplify synaptic learning processes in spite of strong statistical dependencies. Finally we have shown that Hebbian learning also suffices

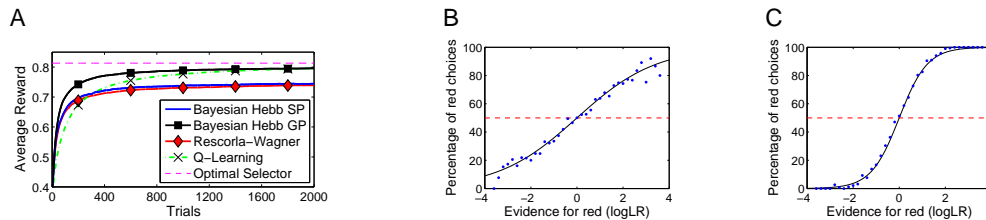

Figure 3: **A)** On 250 4-action conditioning tasks with stochastic rewards, the reward-modulated Bayesian Hebb rule with simple preprocessing (*SP*) learns similarly as the Rescorla-Wagner rule, and substantially faster than Q-learning. With generalized preprocessing (*GP*), the rule converges to the optimal action-selection policy. **B, C)** Action selection policies learned by the reward-modulated Bayesian Hebb rule in the task by Yang and Shadlen [1] after 100 (B), and 1000 (C) trials are qualitatively similar to the policies adopted by monkeys $H$ and $J$ in [1] after learning.

for simple instances of reinforcement learning. The Bayesian Hebb rule, modulated by a signal related to rewards, enables fast learning of optimal action selection. Experimental results of [1] on reinforcement learning of probabilistic inference in primates can be partially modeled in this way with regard to resulting behaviors.

An attractive feature of the Bayesian Hebb rule is its ability to deal with the addition or removal of input features through the creation or deletion of synaptic connections, since no relearning of weights is required for the other synapses. In contrast to discriminative neural learning rules, our approach is generative, which according to [13] leads to faster generalization. Therefore the learning rule may be viewed as a potential building block for models of the brain as a self-organizing and fast adapting probabilistic inference machine.

## Acknowledgments

We would like to thank Martin Bachler, Sophie Deneve, Rodney Douglas, Konrad Koerding, Rajesh Rao, and especially Dan Roth for inspiring discussions. Written under partial support by the Austrian Science Fund FWF, project # P17229-N04, project # S9102-N04, and project # FP6-015879 (FACETS) as well as # FP7-216593 (SECO) of the European Union.

## References

[1] T. Yang and M. N. Shadlen. Probabilistic reasoning by neurons. *Nature*, 447:1075–1080, 2007.

[2] R. P. N. Rao. Neural models of Bayesian belief propagation. In K. Doya, S. Ishii, A. Pouget, and R. P. N. Rao, editors, *Bayesian Brain.*, pages 239–267. MIT-Press, 2007.

[3] C. M. Bishop. *Pattern Recognition and Machine Learning*. Springer (New York), 2006.

[4] S. Deneve. Bayesian spiking neurons I, II. *Neural Computation*, 20(1):91–145, 2008.

[5] A. Sandberg, A. Lansner, K. M. Petersson, and Ö. Ekeberg. A Bayesian attractor network with incremental learning. *Network: Computation in Neural Systems*, 13:179–194, 2002.

[6] D. Roth. Learning in natural language. In *Proc. of IJCAI*, pages 898–904, 1999.

[7] D. O. Hebb. *The Organization of Behavior*. Wiley, New York, 1949.

[8] D. P. Bertsekas and J.N. Tsitsiklis. *Neuro-Dynamic Programming*. Athena Scientific, 1996.

[9] B. Nessler, M. Pfeiffer, and W. Maass. Journal version. *in preparation*, 2009.

[10] L. P. Sugrue, G. S. Corrado, and W. T. Newsome. Matching behavior and the representation of value in the parietal cortex. *Science*, 304:1782–1787, 2004.

[11] J. S. Ide and F. G. Cozman. Random generation of Bayesian networks. In *Proceedings of the 16th Brazilian Symposium on Artificial Intelligence*, pages 366–375, 2002.

[12] R. A. Rescorla and A. R. Wagner. Classical conditioning II. In A. H. Black and W. F. Prokasy, editors, *A theory of Pavlovian conditioning*, pages 64–99. 1972.

[13] A. Y. Ng and M. I. Jordan. On discriminative vs. generative classifiers. *NIPS*, 14:841–848, 2002.

